# On the Complexity of Linear Prediction: Risk Bounds, Margin Bounds, and Regularization

**Sham M. Kakade**
TTI Chicago
Chicago, IL 60637
sham@tti-c.org

**Karthik Sridharan**
TTI Chicago
Chicago, IL 60637
karthik@tti-c.org

**Ambuj Tewari**
TTI Chicago
Chicago, IL 60637
tewari@tti-c.org

## Abstract

This work characterizes the generalization ability of algorithms whose predictions are linear in the input vector. To this end, we provide sharp bounds for Rademacher and Gaussian complexities of (constrained) linear classes, which directly lead to a number of generalization bounds. This derivation provides simplified proofs of a number of corollaries including: risk bounds for linear prediction (including settings where the weight vectors are constrained by either $L_2$ or $L_1$ constraints), margin bounds (including both $L_2$ and $L_1$ margins, along with more general notions based on relative entropy), a proof of the PAC-Bayes theorem, and upper bounds on $L_2$ covering numbers (with $L_p$ norm constraints and relative entropy constraints). In addition to providing a unified analysis, the results herein provide some of the sharpest risk and margin bounds. Interestingly, our results show that the uniform convergence rates of empirical risk minimization algorithms tightly match the regret bounds of online learning algorithms for linear prediction, up to a constant factor of 2.

## 1  Introduction

Linear prediction is the cornerstone of an extensive number of machine learning algorithms, including SVM's, logistic and linear regression, the lasso, boosting, etc. A paramount question is to understand the generalization ability of these algorithms in terms of the attendant complexity restrictions imposed by the algorithm. For example, for the sparse methods (e.g. regularizing based on $L_1$ norm of the weight vector) we seek generalization bounds in terms of the sparsity level. For margin based methods (e.g. SVMs or boosting), we seek generalization bounds in terms of either the $L_2$ or $L_1$ margins. The focus of this paper is to provide a more unified analysis for methods which use linear prediction.

Given a training set $\{(x_i, y_i)\}_{i=1}^n$, the paradigm is to compute a weight vector $\hat{\mathbf{w}}$ which minimizes the $F$-regularized $\ell$-risk. More specifically,

$$\hat{\mathbf{w}} = \operatorname*{argmin}_{\mathbf{w}} \frac{1}{n}\sum_{i=1}^n \ell(\langle \mathbf{w}, x_i \rangle, y_i) + \lambda F(\mathbf{w}) \tag{1}$$

where $\ell$ is the loss function, $F$ is the regularizer, and $\langle \mathbf{w}, \mathbf{x} \rangle$ is the inner product between vectors $\mathbf{x}$ and $\mathbf{w}$. In a formulation closely related to the dual problem, we have:

$$\hat{\mathbf{w}} = \operatorname*{argmin}_{\mathbf{w}: F(\mathbf{w}) \leq c} \frac{1}{n}\sum_{i=1}^n \ell(\langle \mathbf{w}, x_i \rangle, y_i) \tag{2}$$

where, instead of regularizing, a hard restriction over the parameter space is imposed (by the constant $c$). This works provides generalization bounds for an extensive family of regularization functions $F$.

Rademacher complexities (a measure of the complexity of a function class) provide a direct route to obtaining such generalization bounds, and this is the route we take. Such bounds are analogous to VC dimensions bounds, but they are typically much sharper and allow for distribution dependent bounds. There are a number of methods in the literature to use Rademacher complexities to obtain either generalization bounds or margin bounds. Bartlett and Mendelson [2002] provide a generalization bound for Lipschitz loss functions. For binary prediction, the results in Koltchinskii and Panchenko [2002] provide means to obtain margin bounds through Rademacher complexities.

In this work, we provide sharp bounds for Rademacher and Gaussian complexities of linear classes, with respect to a strongly convex complexity function $F$ (as in Equation 1). These bounds provide simplified proofs of a number of corollaries: generalization bounds for the regularization algorithm in Equation 2 (including settings where the weight vectors are constrained by either $L_2$ or $L_1$ constraints), margin bounds (including $L_2$ and $L_1$ margins, and, more generally, for $L_p$ margins), a proof of the PAC-Bayes theorem, and $L_2$ covering numbers (with $L_p$ norm constraints and relative entropy constraints). Our bounds are often tighter than previous results and our proofs are all under this more unified methodology.

Our proof techniques — reminiscent of those techniques for deriving regret bounds for online learning algorithms — are rooted in convex duality (following Meir and Zhang [2003]) and use a more general notion of strong convexity (as in Shalev-Shwartz and Singer [2006]). Interestingly, the risk bounds we provide closely match the regret bounds for online learning algorithms (up to a constant factor of 2), thus showing that the uniform converge rates of empirical risk minimization algorithms tightly match the regret bounds of online learning algorithms (for linear prediction). The Discussion provides this more detailed comparison.

## 1.1 Related Work

A staggering number of results have focused on this problem in varied special cases. Perhaps the most extensively studied are margin bounds for the 0-1 loss. For $L_2$-margins (relevant for SVM's, perceptron based algorithms, etc.), the sharpest bounds are those provided by Bartlett and Mendelson [2002] (using Rademacher complexities) and Langford and Shawe-Taylor [2003], McAllester [2003] (using the PAC-Bayes theorem). For $L_1$-margins (relevant for Boosting, winnow, etc), bounds are provided by Schapire et al. [1998] (using a self-contained analysis) and Langford et al. [2001] (using PAC-Bayes, with a different analysis). Another active line of work is on sparse methods — particularly methods which impose sparsity via $L_1$ regularization (in lieu of the non-convex $L_0$ norm). For $L_1$ regularization, Ng [2004] provides generalization bounds for this case, which follow from the covering number bounds of Zhang [2002]. However, these bounds are only stated as polynomial in the relevant quantities (dependencies are not provided).

Previous to this work, the most unified framework for providing generalization bounds for linear prediction stem from the covering number bounds in Zhang [2002]. Using these covering number bounds, Zhang [2002] derives margin bounds in a variety of cases. However, providing sharp generalization bounds for problems with $L_1$ regularization (or $L_1$ constraints in the dual) requires more delicate arguments. As mentioned, Ng [2004] provides bounds for this case, but the techniques used by Ng [2004] would result in rather loose dependencies (the dependence on the sample size $n$ would be $n^{-1/4}$ rather than $n^{-1/2}$). We discuss this later in Section 4.

## 2 Preliminaries

Our input space, $\mathcal{X}$, is a subset of a vector space, and our output space is $\mathcal{Y}$. Our samples $(X, Y) \in \mathcal{X} \times \mathcal{Y}$ are distributed according to some unknown distribution $P$. The inner product between vectors $\mathbf{x}$ and $\mathbf{w}$ is denoted by $\langle \mathbf{w}, \mathbf{x} \rangle$, where $\mathbf{w} \in S$ (here, $S$ is a subset of the dual space to our input vector space). A norm of a vector $\mathbf{x}$ is denoted by $\|\mathbf{x}\|$, and the dual norm is defined as $\|\mathbf{w}\|_\star = \sup\{\langle \mathbf{w}, \mathbf{x} \rangle : \|\mathbf{x}\| \leq 1\}$. We further assume that for all $\mathbf{x} \in \mathcal{X}$, $\|\mathbf{x}\| \leq X$.

Let $\ell : \mathbf{R} \times \mathcal{Y} \to \mathbb{R}^+$ be our loss function of interest. Throughout we shall consider linear predictors of form $\langle \mathbf{w}, \mathbf{x} \rangle$. The expected of loss of $\mathbf{w}$ is denoted by $\mathcal{L}(\mathbf{w}) = \mathbb{E}[\ell(\langle \mathbf{w}, \mathbf{x} \rangle, y)]$. As usual, we are provided with a sequence of i.i.d. samples $\{(\mathbf{x}_i, y_i)\}_{i=1}^n$, and our goal is to minimize our expected loss. We denote the empirical loss as $\hat{\mathcal{L}}(\mathbf{w}) = \frac{1}{n} \sum_{i=1}^n \ell(\langle \mathbf{w}, \mathbf{x}_i \rangle, y_i)$.

The restriction we make on our complexity function $F$ is that it is a strongly convex function. In particular, we assume it is strongly convex with respect to our dual norm: a function $F : S \to \mathbb{R}$ is said to be $\sigma$-strongly convex w.r.t. to $\|\cdot\|_*$ iff $\forall \mathbf{u}, \mathbf{v} \in S$, $\forall \alpha \in [0,1]$, we have

$$F(\alpha\mathbf{u} + (1-\alpha)\mathbf{v}) \leq \alpha F(\mathbf{u}) + (1-\alpha)F(\mathbf{v}) - \frac{\sigma}{2}\alpha(1-\alpha)\|\mathbf{u}-\mathbf{v}\|_*^2 \,.$$

See Shalev-Shwartz and Singer [2006] for more discussion on this generalized definition of strong convexity.

Recall the definition of the Rademacher and Gaussian complexity of a function class $\mathcal{F}$,

$$\mathcal{R}_n(\mathcal{F}) = \mathbb{E}\left[\sup_{f\in\mathcal{F}} \frac{1}{n}\sum_{i=1}^n f(\mathbf{x}_i)\epsilon_i\right] \qquad \mathcal{G}_n(\mathcal{F}) = \mathbb{E}\left[\sup_{f\in\mathcal{F}} \frac{1}{n}\sum_{i=1}^n f(\mathbf{x}_i)\epsilon_i\right]$$

where, in the former, $\epsilon_i$ independently takes values in $\{-1, +1\}$ with equal probability, and, in the latter, $\epsilon_i$ are independent, standard normal random variables. In both expectations, $(\mathbf{x}_1, \ldots, \mathbf{x}_n)$ are i.i.d.

As mentioned in the Introduction, there are number of methods in the literature to use Rademacher complexities to obtain either generalization bounds or margin bounds. Two results are particularly useful to us. First, Bartlett and Mendelson [2002] provides the following generalization bound for Lipschitz loss functions. Here, $\mathcal{L}(f) = \mathbb{E}[\ell(f(x), y)]$ is the expected of loss of $f : \mathcal{X} \to \mathbb{R}$, and $\hat{\mathcal{L}}(f) = \frac{1}{n}\sum_{i=1}^n \ell(f(x_i), y_i)$ is the empirical loss.

**Theorem 1. (Bartlett and Mendelson [2002])** *Assume the loss $\ell$ is Lipschitz (with respect to its first argument) with Lipschitz constant $L_\ell$ and that $\ell$ is bounded by $c$. For any $\delta > 0$ and with probability at least $1 - \delta$ simultaneously for all $f \in \mathcal{F}$, we have that*

$$\mathcal{L}(f) \leq \hat{\mathcal{L}}(f) + 2L_\ell \mathcal{R}_n(\mathcal{F}) + c\sqrt{\frac{\log(1/\delta)}{2n}}$$

*where $\mathcal{R}_n(\mathcal{F})$ is the Rademacher complexity of a function class $\mathcal{F}$, and $n$ is the sample size.*

The second result, for binary prediction, from Koltchinskii and Panchenko [2002] provides a margin bound in terms of the Rademacher complexity. The following is a variant of Theorem 2 in Koltchinskii and Panchenko [2002]:

**Theorem 2. (Koltchinskii and Panchenko [2002])** *The zero-one loss function is given by $\ell(f(x), y) = \mathbf{1}[yf(x) \leq 0]$, where $y \in \{+1, -1\}$. Denote the fraction of the data having $\gamma$-margin mistakes by $K_\gamma(f) := \frac{|\{i: y_i f(\mathbf{x}_i) < \gamma\}|}{n}$. Assume that $\forall f \in \mathcal{F}$ we have $\sup_{\mathbf{x}\in\mathcal{X}} |f(\mathbf{x})| \leq C$. Then, with probability at least $1 - \delta$ over the sample, for all margins $\gamma > 0$ and all $f \in \mathcal{F}$ we have,*

$$\mathcal{L}(f) \leq K_\gamma(f) + 4\frac{\mathcal{R}_n(\mathcal{F})}{\gamma} + \sqrt{\frac{\log(\log_2\frac{4C}{\gamma})}{n}} + \sqrt{\frac{\log(1/\delta)}{2n}} \,.$$

(We provide a proof in the appendix.) The above results show that if we provide sharp bounds on the Rademacher complexities then we obtain sharp generalization bounds. Typically, we desire upper bounds on the Rademacher complexity that decrease with $n$.

## 3 Complexities of Linear Function Classes

Given a subset $\mathcal{W} \subseteq S$, define the associated class of linear functions $\mathcal{F}_{\mathcal{W}}$ as $\mathcal{F}_{\mathcal{W}} := \{\mathbf{x} \mapsto \langle \mathbf{w}, \mathbf{x}\rangle : \mathbf{w} \in \mathcal{W}\}$. Our main theorem bounds the complexity of $\mathcal{F}_{\mathcal{W}}$ for certain sets $\mathcal{W}$.

**Theorem 3.** *(Complexity Bounds) Let $S$ be a closed convex set and let $F : S \to \mathbb{R}$ be $\sigma$-strongly convex w.r.t. $\|\cdot\|_*$ s.t. $\inf_{\mathbf{w}\in S} F(\mathbf{w}) = 0$. Further, let $\mathcal{X} = \{\mathbf{x} : \|\mathbf{x}\| \leq X\}$. Define $\mathcal{W} = \{\mathbf{w} \in S : F(\mathbf{w}) \leq W_*^2\}$. Then, we have*

$$\mathcal{R}_n(\mathcal{F}_{\mathcal{W}}) \leq XW_*\sqrt{\frac{2}{\sigma n}} \qquad, \qquad \mathcal{G}_n(\mathcal{F}_{\mathcal{W}}) \leq XW_*\sqrt{\frac{2}{\sigma n}} \,.$$

The restriction $\inf_{\mathbf{w}\in S} F(\mathbf{w}) = 0$ is not a significant one since adding a constant to $F$ still keeps it strongly convex. Interestingly, the complexity bounds above precisely match the regret bounds for online learning algorithms (for linear prediction), a point which we return to in the Discussion. We first provide a few examples, before proving this result.

## 3.1 Examples

**(1) $L_p/L_q$ norms.** Let $S = \mathbb{R}^d$. Take $\|\cdot\|, \|\cdot\|_*$ to be the $L_p$, $L_q$ norms for $p \in [2, \infty)$, $1/p + 1/q = 1$, where $\|\mathbf{x}\|_p := \left( \sum_{j=1}^d |\mathbf{x}_i|^p \right)^{1/p}$. Choose $F(\mathbf{w}) = \|\cdot\|_q^2$ and note that it is $2(q-1)$-strongly convex on $\mathbb{R}^d$ w.r.t. itself. Set $\mathcal{X}, \mathcal{W}$ as in Theorem 3. Then, we have

$$\mathcal{R}_n(\mathcal{F}_\mathcal{W}) \leq XW_* \sqrt{\frac{p-1}{n}} \ . \tag{3}$$

**(2) $L_\infty/L_1$ norms.** Let $S = \{\mathbf{w} \in \mathbb{R}^d \ : \ \|\mathbf{w}\|_1 = W_1 \ , \ \mathbf{w}_j \geq 0\}$ be the $W_1$-scaled probability simplex. Take $\|\cdot\|, \|\cdot\|_*$ to be the $L_\infty$, $L_1$ norms, $\|\mathbf{x}\|_\infty = \max_{1 \leq j \leq d} |\mathbf{x}_j|$. Fix a probability distribution $\mu > 0$ and let $F(\mathbf{w}) = \text{entro}_\mu(\mathbf{w}) := \sum_j (\mathbf{w}_j/W_1) \log(\mathbf{w}_j/(W_1 \mu_j))$. For any $\mu$, $\text{entro}_\mu(\mathbf{w})$ is $1/W_1^2$-strongly convex on $S$ w.r.t. $\|\cdot\|_1$. Set $\mathcal{X}$ as in Theorem 3 and let $\mathcal{W}(E) = \{\mathbf{w} \in S \ : \ \text{entro}_\mu(\mathbf{w}) \leq E\}$. Then, we have

$$\mathcal{R}_n(\mathcal{F}_{\mathcal{W}(E)}) \leq XW_1 \sqrt{\frac{2E}{n}} \ . \tag{4}$$

Note that if we take $\mu$ to be the uniform distribution then for any $\mathbf{w} \in S$ we have that trivial upper bound of $\text{entro}_\mu(\mathbf{w}) \leq \log d$. Hence if we let $\mathcal{W} := \mathcal{W}(\log d)$ with uniform $\mu$ and note that it is the entire scaled probability simplex. Then

$$\mathcal{R}_n(\mathcal{F}_\mathcal{W}) \leq XW_1 \sqrt{\frac{2 \log d}{n}} \ . \tag{5}$$

The restriction $\mathbf{w}_j \geq 0$ can be removed in the definition of $S$ by the standard trick of doubling the dimension of $\mathbf{x}$ to include negated copies of each coordinate. So, if we have $S = \{\mathbf{w} \in \mathbb{R}^d \ : \ \|\mathbf{w}\|_1 \leq W_1\}$ and we set $\mathcal{X}$ as above and $\mathcal{W} = S$, then we get $\mathcal{R}_n(\mathcal{F}_\mathcal{W}) \leq XW_1 \sqrt{2 \log(2d)/n}$.

In this way, even though the $L_1$ norm is not strongly convex (so our previous Theorem does not directly apply to it), the class of functions imposed by this $L_1$ norm restriction is equivalent to that imposed by the above entropy restriction. Hence, we are able to analyze the generalization properties of the optimization problem in Equation 2.

**(3) Smooth norms.** A norm is $(2, D)$-smooth on $S$ if for any $\mathbf{x}, \mathbf{y} \in S$,

$$\frac{d^2}{dt^2} \|\mathbf{x} + t\mathbf{y}\|^2 \leq 2D^2 \|\mathbf{y}\|^2 \ .$$

Let $\|\cdot\|$ be a $(2, D)$-smooth norm and $\|\cdot\|_*$ be its dual. Lemma 11 in the appendix proves that $\|\cdot\|_*$ is $2/D^2$-strongly convex w.r.t. itself. Set $\mathcal{X}, \mathcal{W}$ as in Theorem 3. Then, we have

$$\mathcal{R}_n(\mathcal{F}_\mathcal{W}) \leq \frac{XW_* D}{\sqrt{n}} \ . \tag{6}$$

**(4) Bregman divergences.** For a strongly convex $F$, define the *Bregman divergence* $\Delta_F(\mathbf{w}\|\mathbf{v}) := F(\mathbf{w}) - F(\mathbf{v}) - \langle \nabla F(\mathbf{v}), \mathbf{w} - \mathbf{v} \rangle$. It is interesting to note that Theorem 3 is still valid if we choose $W_* = \{\mathbf{w} \in S \ : \ \Delta_F(\mathbf{w}\|\mathbf{v}) \leq W_*^2\}$ for some fixed $\mathbf{v} \in S$. This is because the Bregman divergence $\Delta_F(\cdot\|\mathbf{v})$ inherits the strong convexity of $F$.

Except for (5), none of the above bounds depend explicitly on the dimension of the underlying space and hence can be easily extended to infinite dimensional spaces under appropriate assumptions.

## 3.2 The Proof

First, some background on convex duality is in order. The Fenchel conjugate of $F : S \to \mathbb{R}$ is defined as:

$$F^*(\boldsymbol{\theta}) := \sup_{\mathbf{w} \in S} \langle \mathbf{w}, \boldsymbol{\theta} \rangle - F(\mathbf{w}) \ .$$

A simple consequence of this definition is Fenchel-Young inequality,

$$\forall \boldsymbol{\theta}, \mathbf{w} \in S, \ \langle \mathbf{w}, \boldsymbol{\theta} \rangle \leq F(\mathbf{w}) + F^*(\boldsymbol{\theta}) \ .$$

If $F$ is $\sigma$-strongly convex, then $F^*$ is differentiable and

$$\forall \boldsymbol{\theta}, \boldsymbol{\eta}, \ F^*(\boldsymbol{\theta} + \boldsymbol{\eta}) \leq F^*(\boldsymbol{\theta}) + \langle \nabla F^*(\boldsymbol{\theta}), \boldsymbol{\eta} \rangle + \frac{1}{2\sigma} \|\boldsymbol{\eta}\|_*^2 . \tag{7}$$

See the Appendix in Shalev-Shwartz [2007] for proof. Using this inequality we can control the expectation of $F^*$ applied to a sum of independent random variables.

**Lemma 4.** *Let $S$ be a closed convex set and let $F : S \to \mathbb{R}$ be $\sigma$-strongly convex w.r.t. $\|\cdot\|_*$. Let $Z_i$ be mean zero independent random vectors such that $\mathbb{E}[\|Z_i\|^2] \leq V^2$. Define $S_i := \sum_{j \leq i} Z_i$. Then $F^*(S_i) - iV^2/2\sigma$ is a supermartingale. Furthermore, if $\inf_{\mathbf{w} \in S} F(\mathbf{w}) = 0$, then $\mathbb{E}[F^*(S_n)] \leq nV^2/2\sigma$.*

*Proof.* Note that $\inf_{\mathbf{w} \in S} F(\mathbf{w}) = 0$ implies $F^*(\mathbf{0}) = 0$. Inequality (7) gives,

$$F^*(S_{i-1} + Z_i) \leq F^*(S_i) + \langle \nabla F^*(S_{i-1}), Z_i \rangle + \frac{1}{2\sigma} \|Z_i\|_*^2 .$$

Taking conditional expectation w.r.t. $Z_1, \ldots, Z_{i-1}$ and noting that $\mathbb{E}_{i-1}[Z_i] = 0$ and $\mathbb{E}_{i-1}[\|Z_i\|_*^2] \leq V^2$, we get

$$\mathbb{E}_{i-1}[F^*(S_i)] \leq F^*(S_{i-1}) + 0 + \frac{V^2}{2\sigma}$$

where $\mathbb{E}_{i-1}[\cdot]$ abbreviates $\mathbb{E}[\cdot \mid Z_1, \ldots, Z_{i-1}]$. To end the proof, note that $\inf_{\mathbf{w} \in S} F(\mathbf{w}) = 0$ implies $F^*(\mathbf{0}) = 0$. $\square$

Like Meir and Zhang [2003] (see Section 5 therein), we begin by using conjugate duality to bound the Rademacher complexity. To finish the proof, we exploit the strong convexity of $F$ by applying the above lemma.

*Proof.* Fix $\mathbf{x}_1, \ldots, \mathbf{x}_n$ such that $\|\mathbf{x}_i\| \leq X$. Let $\boldsymbol{\theta} = \frac{1}{n} \sum_i \epsilon_i \mathbf{x}_i$ where $\epsilon_i$'s are i.i.d. Rademacher or Gaussian random variables (our proof only requires that $\mathbb{E}[\epsilon_i] = 0$ and $\mathbb{E}[\epsilon_i^2] = 1$). Choose arbitrary $\lambda > 0$. By Fenchel's inequality, we have $\langle \mathbf{w}, \lambda \boldsymbol{\theta} \rangle \leq F(\mathbf{w}) + F^*(\lambda \boldsymbol{\theta})$ which implies

$$\langle \mathbf{w}, \boldsymbol{\theta} \rangle \leq \frac{F(\mathbf{w})}{\lambda} + \frac{F^*(\lambda \boldsymbol{\theta})}{\lambda} .$$

Since, $F(\mathbf{w}) \leq W_*^2$ for all $\mathbf{w} \in \mathcal{W}$, we have

$$\sup_{\mathbf{w} \in \mathcal{W}} \langle \mathbf{w}, \boldsymbol{\theta} \rangle \leq \frac{W_*^2}{\lambda} + \frac{F^*(\lambda \boldsymbol{\theta})}{\lambda} .$$

Taking expectation (w.r.t. $\epsilon_i$'s), we get

$$\mathbb{E}\left[ \sup_{\mathbf{w} \in \mathcal{W}} \langle \mathbf{w}, \boldsymbol{\theta} \rangle \right] \leq \frac{W_*^2}{\lambda} + \frac{1}{\lambda} \mathbb{E}\left[ F^*(\lambda \boldsymbol{\theta}) \right] .$$

Now set $Z_i = \frac{\lambda \epsilon_i \mathbf{x}_i}{n}$ (so that $S_n = \lambda \boldsymbol{\theta}$) and note that the conditions of Lemma 4 are satisfied with $V^2 = \lambda^2 B^2 / n^2$ and hence $\mathbb{E}[F^*(\lambda \boldsymbol{\theta})] \leq \frac{\lambda^2 X^2}{2\sigma n}$. Plugging this above, we have

$$\mathbb{E}\left[ \sup_{\mathbf{w} \in \mathcal{W}} \langle \mathbf{w}, \boldsymbol{\theta} \rangle \right] \leq \frac{W_*^2}{\lambda} + \frac{\lambda X^2}{2\sigma n} .$$

Setting $\lambda = \sqrt{\frac{2\sigma n W_*^2}{X^2}}$ gives

$$\mathbb{E}\left[ \sup_{\mathbf{w} \in \mathcal{W}} \langle \mathbf{w}, \boldsymbol{\theta} \rangle \right] \leq X W_* \sqrt{\frac{2}{\sigma n}} .$$

which completes the proof. $\square$

## 4 Corollaries

### 4.1 Risk Bounds

We now provide generalization error bounds for any Lipschitz loss function $\ell$, with Lipschitz constant $L_\ell$. Based on the Rademacher generalization bound provided in the Introduction (see Theorem 1) and the bounds on Rademacher complexity proved in previous section, we obtain the following corollaries.

**Corollary 5.** *Each of the following statements holds with probability at least $1 - \delta$ over the sample:*

- *Let $\mathcal{W}$ be as in the $L_p/L_q$ **norms** example. For all $\mathbf{w} \in \mathcal{W}$,*

$$\mathcal{L}(\mathbf{w}) \le \hat{\mathcal{L}}(\mathbf{w}) + 2L_\ell X W_* \sqrt{\frac{p-1}{n}} + L_\ell X W_* \sqrt{\frac{\log(1/\delta)}{2n}}$$

- *Let $\mathcal{W}$ be as in the $L_\infty/L_1$ **norms** example. For all $\mathbf{w} \in \mathcal{W}$,*

$$\mathcal{L}(\hat{\mathbf{w}}) \le \hat{\mathcal{L}}(\mathbf{w}) + 2L_\ell X W_1 \sqrt{\frac{2\log(d)}{n}} + L_\ell X W_1 \sqrt{\frac{\log(1/\delta)}{2n}}$$

Ng [2004] provides bounds for methods which use $L_1$ regularization. These bounds are only stated as polynomial bounds, and, the methods used (covering number techniques from Pollard [1984] and covering number bounds from Zhang [2002]) would provide rather loose bounds (the $n$ dependence would be $n^{-1/4}$). In fact, even a more careful analysis via Dudley's entropy integral using the covering numbers from Zhang [2002] would result in a worse bound (with additional $\log n$ factors). The above argument is sharp and rather direct.

### 4.2 Margin Bounds

In this section we restrict ourselves to binary classification where $\mathcal{Y} = \{+1, -1\}$. Our prediction is given by $\mathrm{sign}(\langle \mathbf{w}, \mathbf{x} \rangle)$. The zero-one loss function is given by $\ell(\langle \mathbf{w}, \mathbf{x} \rangle, y) = \mathbf{1}[y \langle \mathbf{w}, \mathbf{x} \rangle \le 0]$. Denote the fraction of the data having $\gamma$-margin mistakes by $K_\gamma(f) := \frac{|\{i : y_i f(\mathbf{x}_i) < \gamma\}|}{n}$. We now demonstrate how to get improved margin bounds using the upper bounds for the Rademacher complexity derived in Section 3.

Based on the Rademacher margin bound provided in the Introduction (see Theorem 2), we get the following corollary which will directly imply the margin bounds we are aiming for. The bound for the $p = 2$ case has been used to explain the performance of SVMs. Our bound essentially matches the best known bound [Bartlett and Mendelson, 2002] which was an improvement over previous bounds [Bartlett and Shawe-Taylor, 1999] proved using fat-shattering dimension estimates. For the $L_\infty/L_1$ case, our bound improves the best known bound [Schapire et al., 1998] by removing a factor of $\sqrt{\log n}$.

**Corollary 6.** *($L_p$ Margins) Each of the following statements holds with probability at least $1 - \delta$ over the sample:*

- *Let $\mathcal{W}$ be as in the $L_p/L_q$ **norms** example. For all $\gamma > 0$, $\mathbf{w} \in \mathcal{W}$,*

$$\mathcal{L}(\mathbf{w}) \le K_\gamma(\mathbf{w}) + 4\frac{XW_*}{\gamma}\sqrt{\frac{p-1}{n}} + \sqrt{\frac{\log(\log_2 \frac{4XW_*}{\gamma})}{n}} + \sqrt{\frac{\log(1/\delta)}{2n}}$$

- *Let $\mathcal{W}$ be as in the $L_\infty/L_1$ **norms** example. For all $\gamma > 0$, $\mathbf{w} \in \mathcal{W}$,*

$$\mathcal{L}(\mathbf{w}) \le K_\gamma(\mathbf{w}) + 4\frac{XW_1}{\gamma}\sqrt{\frac{2\log(d)}{n}} + \sqrt{\frac{\log(\log_2 \frac{4XW_1}{\gamma})}{n}} + \sqrt{\frac{\log(1/\delta)}{2n}}$$

The following result improves the best known results of the same kind, [Langford et al., 2001, Theorem 5] and [Zhang, 2002, Theorem 7], by removing a factor of $\sqrt{\log n}$. These results themselves were an improvement over previous results obtained using fat-shattering dimension estimates.

**Corollary 7.** *(Entropy Based Margins) Let $\mathcal{X}$ be such that for all $\mathbf{x} \in \mathcal{X}$, $\|\mathbf{x}\|_\infty \leq X$. Consider the class $\mathcal{W} = \{w \in \mathbb{R}^d : \|\mathbf{w}\|_1 \leq W_1\}$. Fix an arbitrary prior $\mu$. We have that with probability at least $1 - \delta$ over the sample, for all margins $\gamma > 0$ and all weight vector $\mathbf{w} \in \mathcal{W}$,*

$$\mathcal{L}(\mathbf{w}) \leq K_\gamma(\mathbf{w}) + 8.5 \, \frac{XW_1}{\gamma} \sqrt{\frac{\mathrm{entro}_\mu(\mathbf{w}) + 2.5}{n}} + \sqrt{\frac{\log(\log_2 \frac{4XW_1}{\gamma})}{n}} + \sqrt{\frac{\log(1/\delta)}{2n}}$$

*where* $\mathrm{entro}_\mu(\mathbf{w}) := \sum_i \frac{|\mathbf{w}_i|}{\|\mathbf{w}\|_1} \log(\frac{|\mathbf{w}_i|}{\mu_i \|\mathbf{w}\|_1})$

*Proof.* Proof is provided in the appendix. $\qquad\square$

## 4.3   PAC-Bayes Theorem

We now show that (a form of) the PAC Bayesian theorem [McAllester, 1999] is a consequence of Theorem 3. In the PAC Bayesian theorem, we have a set of hypothesis (possibly infinite) $\mathcal{C}$. We choose some prior distribution over this hypothesis set say $\mu$, and after observing the training data, we choose any arbitrary posterior $\nu$ and the loss we are interested in is $\ell_\nu(\mathbf{x}, y) = \mathbb{E}_{c \sim \nu} \ell(c, \mathbf{x}, y)$ that is basically the expectation of the loss when hypothesis $c \in \mathcal{C}$ are drawn i.i.d. using distribution $\nu$. Note that in this section we are considering a more general form of the loss.

The key observation as that we can view $\ell_\nu(\mathbf{x})$ as the inner product $\langle d\nu(\cdot), \ell(\cdot, \mathbf{x}, y) \rangle$ between the measure $d\nu(\cdot)$ and the loss $\ell(\cdot, x)$. This leads to the following straightforward corollary.

**Corollary 8.** *(PAC-Bayes) For a fixed prior $\mu$ over the hypothesis set $\mathcal{C}$, and any loss bounded by $1$, with probability at least $1 - \delta$ over the sample, simultaneously for all choice of posteriors $\nu$ over $\mathcal{C}$ we have that,*

$$\mathcal{L}_\nu \leq \hat{\mathcal{L}}_\nu + 4.5 \sqrt{\frac{\max\{\mathrm{KL}(\nu\|\mu), 2\}}{n}} + \sqrt{\frac{\log(1/\delta)}{2n}} \qquad (8)$$

*Proof.* Proof is provided in the appendix. $\qquad\square$

Interestingly, this result is an improvement over the original statement, in which the last term was $\sqrt{\log(n/\delta)/n}$. Our bound removes this extra $\log(n)$ factor, so, in the regime where we fix $\nu$ and examine large $n$, this bound is sharper. We note that our goal was not to prove the PAC-Bayes theorem, and we have made little attempt to optimize the constants.

## 4.4   Covering Number Bounds

It is worth noting that using Sudakov's minoration results we can obtain upper bound on the $L_2$ (and hence also $L_1$) covering numbers using the Gaussian complexities. The following is a direct corollary of the Sudakov minoration theorem for Gaussian complexities (Theorem 3.18, Page 80 of Ledoux and Talagrand [1991]).

**Corollary 9.** *Let $\mathcal{F}_\mathcal{W}$ be the function class from Theorem 3. There exists a universal constant $K > 0$ such that its $L_2$ covering number is bounded as follows:*

$$\forall \epsilon > 0 \quad \log(\mathcal{N}_2(\mathcal{F}_\mathcal{W}, \epsilon, n)) \leq \frac{2K^2 X^2 W_*^2}{\sigma \epsilon^2}$$

This bound is sharper than those that could be derived from the $\mathcal{N}_\infty$ covering number bounds of Zhang [2002].

## 5   Discussion: Relations to Online, Regret Minimizing, Algorithms

In this section, we make a further assumption that loss $\ell(\langle \mathbf{w}, \mathbf{x} \rangle, y)$ is convex in its first argument. We now show that in the online setting that the regret bounds for linear prediction closely match our risk bounds. The algorithm we consider performs the update,

$$\mathbf{w}_{t+1} = \nabla F^{-1}(\nabla F(\mathbf{w}_t) - \eta \nabla_\mathbf{w} \ell(\langle \mathbf{w}_t, \mathbf{x}_t \rangle, y_t)) \qquad (9)$$

This algorithm captures both gradient updates, multiplicative updates, and updates based on the $L_p$ norms, through appropriate choices of $F$. See Shalev-Shwartz [2007] for discussion.

For the algorithm given by the above update, the following theorem is a bound on the cumulative regret. It is a corollary of Theorem 1 in Shalev-Shwartz and Singer [2006] (and also of Corollary 1 in Shalev-Shwartz [2007]), applied to our linear case.

**Corollary 10.** *(Shalev-Shwartz and Singer [2006]) Let $S$ be a closed convex set and let $F : S \to \mathbb{R}$ be $\sigma$-strongly convex w.r.t. $\| \cdot \|_*$. Further, let $\mathcal{X} = \{\mathbf{x} \ : \ \|\mathbf{x}\| \leq X\}$ and $\mathcal{W} = \{\mathbf{w} \in S \ : \ F(\mathbf{w}) \leq W_*^2\}$. Then for the update given by Equation 9 if we start with $\mathbf{w}_1 = \arg\min F(\mathbf{w})$, we have that for all sequences $\{(\mathbf{x}_t, y_t)\}_{t=1}^n$,*

$$\sum_{t=1}^n \ell(\langle \mathbf{w}_t, \mathbf{x}_t \rangle, y_t) - \operatorname*{argmin}_{\mathbf{w} \in \mathcal{W}} \sum_{t=1}^n \ell(\langle \mathbf{w}, \mathbf{x}_t \rangle, y_t) \leq L_\ell X W_* \sqrt{\frac{2n}{\sigma}}$$

For completeness, we provide a direct proof in the Appendix. Interestingly, the regret above is precisely our complexity bounds (when $L_\ell = 1$). Also, our risk bounds are a factor of 2 worse, essentially due to the symmetrization step used in proving Theorem 1.

# References

P. L. Bartlett and S. Mendelson. Rademacher and Gaussian complexities: Risk bounds and structural results. *Journal of Machine Learning Research*, 3:463–482, 2002.

P. L. Bartlett and J. Shawe-Taylor. Generalization performance of support vector machines and other pattern classifiers. In B. Schölkopf, C. J. C. Burges, and A. J. Smola, editors, *Advances in Kernel Methods – Support Vector Learning*, pages 43–54. MIT Press, 1999.

N. Cesa-Bianchi and G. Lugosi. *Prediction, learning, and games*. Cambridge University Press, 2006.

V. Koltchinskii and D. Panchenko. Empirical margin distributions and bounding the generalization error of combined classifiers. *Annals of Statistics*, 30(1):1–50, 2002.

J. Langford and J. Shawe-Taylor. PAC-Bayes & margins. In *Advances in Neural Information Processing Systems 15*, pages 423–430, 2003.

J. Langford, M. Seeger, and Nimrod Megiddo. An improved predictive accuracy bound for averaging classifiers. In *Proceedings of the Eighteenth International Conference on Machine Learning*, pages 290–297, 2001.

M. Ledoux and M. Talagrand. *Probability in Banach spaces: Isoperimetry and processes*, volume 23 of *Ergebnisse der Mathematik und ihrer Grenzgebiete (3)*. Springer-Verlag, 1991.

David A. McAllester. Simplified PAC-Bayesian margin bounds. In *Proceedings of the Sixteenth Annual Conference on Computational Learning Theory*, pages 203–215, 2003.

David A. McAllester. PAC-Bayesian model averaging. In *Proceedings of the Twelfth Annual Conference on Computational Learning Theory*, pages 164–170, 1999.

Ron Meir and Tong Zhang. Generalization error bounds for Bayesian mixture algorithms. *Journal of Machine Learning Research*, 4:839–860, 2003.

A.Y. Ng. Feature selection, $l_1$ vs. $l_2$ regularization, and rotational invariance. In *Proceedings of the Twenty-First International Conference on Machine Learning*, 2004.

David Pollard. *Convergence of Stochastic Processes*. Springer-Verlag, 1984.

R.E. Schapire, Y. Freund, P. Bartlett, and W.S. Lee. Boosting the margin: A new explanation for the effectiveness of voting methods. *The Annals of Statistics*, 26(5):1651–1686, October 1998.

S. Shalev-Shwartz. *Online Learning: Theory, Algorithms, and Applications*. PhD thesis, The Hebrew University, 2007.

S. Shalev-Shwartz and Y. Singer. Convex repeated games and Fenchel duality. In *Advances in Neural Information Processing Systems 20*, 2006.

M. Warmuth and A. K. Jagota. Continuous versus discrete-time non-linear gradient descent: Relative loss bounds and convergence. In *Fifth International Symposium on Artificial Intelligence and Mathematics*, 1997.

T. Zhang. Covering number bounds of certain regularized linear function classes. *Journal of Machine Learning Research*, 2:527–550, 2002.
